# SKELETONIZATION:
# A TECHNIQUE FOR TRIMMING THE FAT
# FROM A NETWORK VIA RELEVANCE ASSESSMENT

Michael C. Mozer
Paul Smolensky
Department of Computer Science &
Institute of Cognitive Science
University of Colorado
Boulder, CO 80309-0430

## ABSTRACT

This paper proposes a means of using the knowledge in a network to determine the functionality or *relevance* of individual units, both for the purpose of understanding the network's behavior and improving its performance. The basic idea is to iteratively train the network to a certain performance criterion, compute a measure of relevance that identifies which input or hidden units are most critical to performance, and automatically trim the least relevant units. This *skeletonization* technique can be used to simplify networks by eliminating units that convey redundant information; to improve learning performance by first learning with spare hidden units and then trimming the unnecessary ones away, thereby constraining generalization; and to understand the behavior of networks in terms of minimal "rules."

## INTRODUCTION

One thing that connectionist networks have in common with brains is that if you open them up and peer inside, all you can see is a big pile of goo. Internal organization is obscured by the sheer number of units and connections. Although techniques such as hierarchical cluster analysis (Sejnowski & Rosenberg, 1987) have been suggested as a step in understanding network behavior, one would like a better handle on the role that individual units play. This paper proposes one means of using the knowledge in a network to determine the functionality or *relevance* of individual units. Given a measure of relevance for each unit, the least relevant units can be automatically trimmed from the network to construct a *skeleton* version of the network.

Skeleton networks have several potential applications:

- *Constraining generalization.* By eliminating input and hidden units that serve no purpose, the number of parameters in the network is reduced and generalization will be constrained (and hopefully improved).
- *Speeding up learning.* Learning is fast with many hidden units, but a large number of hidden units allows for many possible generalizations. Learning is slower with few

hidden units, but generalization tends to be better. One idea for speeding up learning is to train a network with many hidden units and then eliminate the irrelevant ones. This may lead to a rapid learning of the training set, and then gradually, an improvement in generalization performance.

- *Understanding the behavior of a network in terms of "rules"*. One often wishes to get a handle on the behavior of a network by analyzing the network in terms of a small number of rules instead of an enormous number of parameters. In such situations, one may prefer a simple network that performed correctly on 95% of the cases over a complex network that performed correctly on 100%. The skeletonization process can discover such a simplified network.

Several researchers (Chauvin, 1989; Hanson & Pratt, 1989; David Rumelhart, personal communication, 1988) have studied techniques for the closely related problem of reducing the number of free parameters in back propagation networks. Their approach involves adding extra cost terms to the usual error function that cause nonessential weights and units to decay away. We have opted for a different approach — the all-or-none removal of units — which is not a gradient descent procedure. The motivation for our approach was twofold. First, our initial interest was in designing a procedure that could serve to focus "attention" on the most important units, hence an explicit relevance metric was needed. Second, our impression is that it is a tricky matter to balance a primary and secondary error term against one another. One must determine the relative weighting of these terms, weightings that may have to be adjusted over the course of learning. In our experience, it is often impossible to avoid local minima — compromise solutions that partially satisfy each of the error terms. This conclusion is supported by the experiments of Hanson and Pratt (1989).

## DETERMINING THE RELEVANCE OF A UNIT

Consider a multi-layer feedforward network. How might we determine whether a given unit serves an important function in the network? One obvious source of information is its outgoing connections. If a unit in layer $l$ has many large-weighted connections, then one might expect its activity to have a big impact on higher layers. However, this need not be. The effects of these connections may cancel each other out; even a large input to units in layer $l+1$ will have little influence if these units are near saturation; outgoing connections from the innervated units in $l+1$ may be small; and the unit in $l$ may have a more-or-less constant activity, in which case it could be replaced by a bias on units in $l+1$. Thus, a more accurate measure of the relevance of a unit is needed.

What one really wants to know is, what will happen to the performance of the network when a unit is removed? That is, how well does the network do *with* the unit versus *without* it? For unit $i$, then, a straightforward measure of the relevance, $\rho_i$, is

$$\rho_i = E_{without\ unit\ i} - E_{with\ unit\ i} ,$$

where $E$ is the error of the network on the training set. The problem with this measure is that to compute the error with a given unit removed, a complete pass must be made through the training set. Thus, the cost of computing $\rho$ is $O(np)$ stimulus presentations, where $n$ is the number of units in the network and $p$ is the number of patterns in the

training set. Further, if the training set is not fixed or is not known to the experimenter, additional difficulties arise in computing $\rho$.

We therefore set out to find a good approximation to $\rho$. Before presenting this approximation, it is first necessary to introduce an additional bit of notation. Suppose that associated with each unit $i$ is a coefficient $\alpha_i$ which represents the *attentional strength* of the unit (see Figure 1). This coefficient can be thought of as gating the flow of activity from the unit:

$$o_j = f\left(\sum_i w_{ji}\,\alpha_i\,o_i\right),$$

where $o_j$ is the activity of unit $j$, $w_{ji}$ the connection strength to $j$ from $i$, and $f$ the sigmoid squashing function. If $\alpha_i = 0$, unit $i$ has no influence on the rest of the network; if $\alpha_i = 1$, unit $i$ is a conventional unit. In terms of $\alpha$, the relevance of unit $i$ can then be rewritten as

$$\rho_i = E_{\alpha_i=0} - E_{\alpha_i=1}\,.$$

We can approximate $\rho_i$ using the derivative of the error with respect to $\alpha$:

$$\lim_{\gamma\to 1}\frac{E_{\alpha_i=\gamma} - E_{\alpha_i=1}}{\gamma - 1} = \left.\frac{\partial E}{\partial \alpha_i}\right|_{\alpha_i=1}$$

Assuming that this equality holds approximately for $\gamma = 0$:

$$\frac{E_{\alpha_i=0} - E_{\alpha_i=1}}{-1} \approx \left.\frac{\partial E}{\partial \alpha_i}\right|_{\alpha_i=1} \qquad \text{or} \qquad -\rho_i \approx \left.\frac{\partial E}{\partial \alpha_i}\right|_{\alpha_i=1}$$

Our approximation for $\rho_i$ is then $\hat{\rho}_i = -\dfrac{\partial E}{\partial \alpha_i}$.

This derivative can be computed using an error propagation procedure very similar to that used in adjusting the weights with back propagation. Additionally, note that because the approximation assumes that $\alpha_i$ is 1, the $\alpha_i$ never need be changed. Thus, the $\alpha_i$ are not actual parameters of the system, just a bit of notational convenience used in

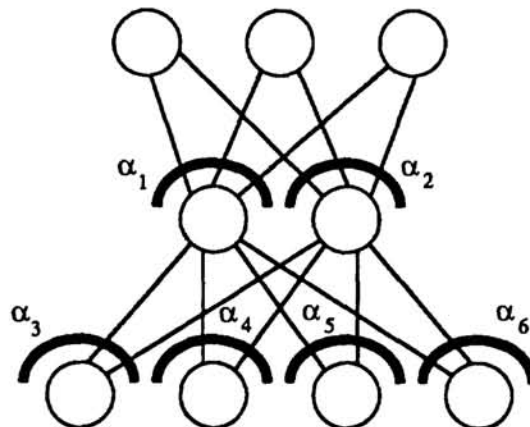

Figure 1. A 4-2-3 network with attentional coefficients on the input and hidden units.

estimating relevance.

In practice, we have found that $\partial E /\partial \alpha$ fluctuates strongly in time and a more stable estimate that yields better results is an exponentially-decaying time average of the derivative. In the simulations reported below, we use the following measure:

$$\rho_i(t+1) = .8\rho_i(t) + .2\frac{\partial E(t)}{\partial \alpha_i} .$$

One final detail of relevance assessment we need to mention is that relevance is computed based on a linear error function, $E^l = \sum |t_{pj} - o_{pj}|$ (where $p$ is an index over patterns, $j$ over output units; $t_{pj}$ is the target output, $o_{pj}$ the actual output). The usual quadratic error function, $E^q = \sum(t_{pj} - o_{pj})^2$, provides a poor estimate of relevance if the output pattern is close to the target. This difficulty with $E^q$ is further elaborated in Mozer and Smolensky (1989). In the results reported below, while $E^q$ is used as the error metric in training the weights via conventional back propagation, $\rho$ is measured using $E^l$. This involves separate back propagation phases for computing the weight updates and the relevance measures.

## A SIMPLE EXAMPLE: THE CUE SALIENCE PROBLEM

Consider a network with four inputs labeled A-D, one hidden unit, and one output. We generated ten training patterns such that the correlations between each input unit and the output are as shown in the first row of Table 1. (In this particular task, a hidden layer is not necessary. The inclusion of the hidden unit simply allowed us to use a standard three-layer architecture for all tasks.)

In this and subsequent simulations, unit activities range from −1 to 1, input and target output patterns are binary (−1 or 1) vectors. Training continues until all output activities are within some acceptable margin of the target value. Additional details of the training procedure and network parameters are described in Mozer and Smolensky (1989).

To perform perfectly, the network need only attend to input A. This is not what the input-hidden connections do, however; their weights have the same qualitative profile as the correlations (second row of Table 1).[1] In contrast, the relevance values for the input

Table 1

| | Input Unit | | | |
| --- | --- | --- | --- | --- |
| | A | B | C | D |
| Correlation with Output Unit | 1.0 | 0.6 | 0.2 | 0.0 |
| Input-Hidden Connection Strengths | 3.15 | 1.23 | .83 | −.01 |
| $\rho_i$ | 5.36 | 0.07 | 0.06 | 0.00 |
| $\rho_i$ | 0.46 | −0.03 | 0.01 | −0.02 |

units show A to be highly relevant while B-D have negligible relevance. Further, the qualitative picture presented by the profile of $\hat{\rho}_i$s is identical to that of the $\rho_i$s. Thus, while the weights merely reflect the statistics of the training set, $\hat{\rho}_i$ indicates the *functionality* of the units.

## THE RULE-PLUS-EXCEPTION PROBLEM

Consider a network with four binary inputs labeled A-D and one binary output. The task is to learn the function AB+$\overline{\text{ABCD}}$; the output unit should be on whenever both A and B are on, or in the special case that all inputs are off. With two hidden units, back propagation arrives at a solution in which one unit responds to AB — the rule — and the other to $\overline{\text{ABCD}}$ — the exception. Clearly, the AB unit is more relevant to the solution; it accounts for fifteen cases whereas the $\overline{\text{ABCD}}$ unit accounts for only one. This fact is reflected in the $\hat{\rho}_i$: in 100 replications of the simulation, the mean value of $\hat{\rho}_{AB}$ was 1.49 whereas $\hat{\rho}_{\overline{ABCD}}$ was only .17. These values are extremely reliable; the standard errors are .003 and .005, respectively.

Relevance was also measured using the quadratic error function. With this metric, the AB unit is incorrectly judged as being *less* relevant than the $\overline{\text{ABCD}}$ unit: $\hat{\rho}_{AB}^q$ is .029 and $\hat{\rho}_{\overline{ABCD}}^q$ is .033. As mentioned above, the basis of the failure of the quadratic error function is that $\hat{\rho}^q$ grossly underestimates the true relevance as the output error goes to zero. Because the one exception pattern is invariably the last to be learned, the output error for the fifteen non-exception patterns is significantly lower, and consequently, the relevance values computed on the basis of the non-exception patterns are much smaller than those computed on the basis of the one exception pattern. This results in the relevance assessment derived from the exception pattern dominating the overall relevance measure, and in the incorrect relevance assignments described above. However, this problem can be avoided by assessing relevance using the linear error function.

If we attempted to "trim" the rule-plus-exception network by eliminating hidden units, the logical first candidate would be the less relevant $\overline{\text{ABCD}}$ unit. This trimming process would leave us with a simpler network — a skeleton network — whose behavior is easily characterized in terms of a simple rule, but which could only account for 15 of the 16 input cases.

## CONSTRUCTING SKELETON NETWORKS

In the remaining examples we construct skeleton networks using the relevance metric. The procedure is as follows: (1) train the network until all output unit activities are within some specified margin around the target value (for details, see Mozer & Smolensky, 1989); (2) compute $\hat{\rho}$ for each unit; (3) remove the unit with the smallest $\hat{\rho}$; and (4) repeat steps 1-3 a specified number of times. In the examples below, we have chosen to trim either the input units or the hidden units, not both simultaneously, but there is no reason why this could not be done.

We have not yet addressed the crucial question of how much to trim away from the network. At present, we specify in advance when to stop trimming. However, the procedure described above makes use only of the ordinal values of the $\hat{\rho}$. One untapped

source of information that may be quite informative is the magnitudes of the $\beta$. A large increase in the minimum $\beta$ value as trimming progresses may indicate that further trimming will seriously disrupt performance in the network.

## THE TRAIN PROBLEM

Consider the task of determining a rule that discriminates the "east" trains from the "west" trains in Figure 2. There are two simple rules — simple in the sense that the rules require a minimal number of input features: East trains have *a long car and triangle load in car* or *an open car or white wheels on car*. Thus, of the seven features that describe each train, only two are essential for making the east/west discrimination.

A 7-1-1 network trained on this task using back propagation learns quickly, but the final solution takes into consideration nearly all the inputs because 6 of the 7 features are partially correlated with the east/west discrimination. When the skeletonization procedure is applied to trim the number of inputs from 7 to 2, however, the network is successfully trimmed to the minimal set of input features — either *long car* and *triangle load*, or *open car* and *white wheels on car* — on each of 100 replications of the simulation we ran.

The trimming task is far from trivial. The expected success rate with random removal of the inputs is only 9.5%. Other skeletonization procedures we experimented with resulted in success rates of 50%-90%.

## THE FOUR-BIT MULTIPLEXOR PROBLEM

Consider a network that learns to behave as a four-bit multiplexor. The task is, given 6 binary inputs labeled A-D, $M_1$, and $M_2$, and one binary output, to map one of the inputs A-D to the output contingent on the values of $M_1$ and $M_2$. The logical function being computed is $\overline{M_1M_2}A + \overline{M_1}M_2B + M_1\overline{M_2}C + M_1M_2D$.

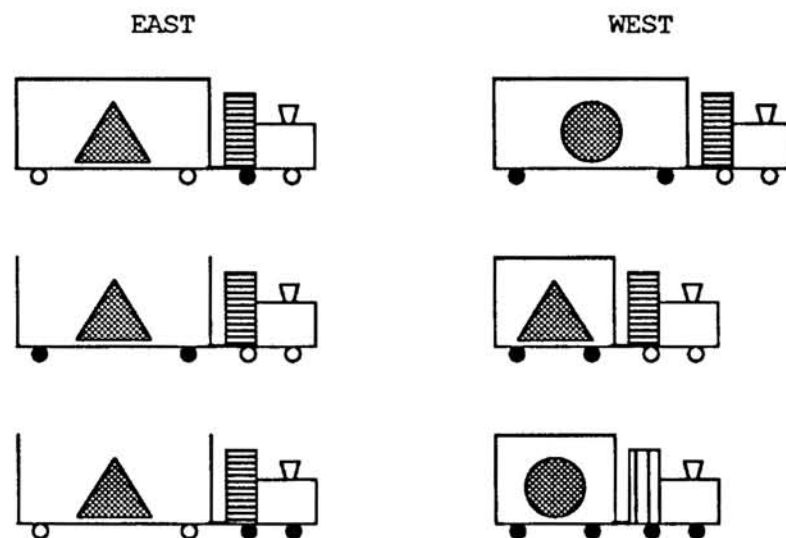

Figure 2. The train problem. Adapted from Medin, Wattenmaker, & Michalski, 1987.

Table 2

| architecture | failure rate | median epochs to criterion (with 8 hidden) | median epochs to criterion (with 4 hidden) |
|---|---|---|---|
| standard 4-hidden net | 17% | -- | 52 |
| 8→4 skeleton net | 0% | 25 | 45 |

A standard 4-hidden unit back propgation network was tested against a skeletonized network that began with 8 hidden units initially and was trimmed to 4 (an 8→4 *skeleton network*). If the network did not reach the performance criterion within 1000 training epochs, we assumed that the network was stuck in a local minimum and counted the run as a failure.

Performance statistics for the two networks are shown in Table 2, averaged over 100 replications. The standard network fails to reach criterion on 17% of the runs, whereas the skeleton network always obtains a solution with 8 hidden units and the solution is not lost as the hidden layer is trimmed to 4 units.[2] The skeleton network with 8 hidden units reaches criterion in about half the number of training epochs required by the standard network. From this point, hidden units are trimmed one at a time from the skeleton network, and after each cut the network is retrained to criterion. Nonetheless, the total number of epochs required to train the initial 8 hidden unit network and then trim it down to 4 is still less than that required for the standard network with 4 units. Furthermore, as hidden units are trimmed, the performance of the skeleton network remains close to criterion, so the improvement in learning is substantial.

## THE RANDOM MAPPING PROBLEM

The problem here is to map a set of random 20-element input vectors to random 2-element output vectors. Twenty random input-output pairs were used as the training set. Ten such training sets were generated and tested. A standard 2-hidden unit network was tested against a 6→2 skeleton network. For each training set and architecture, 100 replications of the simulation were run. If criterion was not reached within 1000 training epochs, we assumed that the network was stuck in a local minimum and counted the run as a failure.

As Table 3 shows, the standard network failed to reach criterion with two hidden units on 17% of all runs, whereas the skeleton network failed with the hidden layer trimmed to two units on only 8.3% of runs. In 9 of the 10 training sets, the failure rate of the skeleton network was lower than that of the standard network. Both networks required comparable amounts of training to reach criterion with two hidden units, but the skeleton network reaches criterion much sooner with six hidden units, and its performance does not significantly decline as the network is trimmed. These results parallel those of the four-bit multiplexor.

Table 3

| training set | standard network | | 6→2 skeleton network | | |
|---|---|---|---|---|---|
| | % failures | median epochs to criterion (2 hidden) | % failures | median epochs to criterion (6 hidden) | median epochs to criterion (2 hidden) |
| 1 | 14 | 20 | 7 | 11 | 22 |
| 2 | 16 | 69 | 13 | 12 | 47 |
| 3 | 25 | 34 | 0 | 7 | 14 |
| 4 | 33 | 38 | 0 | 10 | 21 |
| 5 | 38 | 96 | 55 | 35 | <max> |
| 6 | 9 | 17 | 1 | 9 | 17 |
| 7 | 9 | 28 | 5 | 14 | 43 |
| 8 | 6 | 13 | 0 | 8 | 16 |
| 9 | 8 | 12 | 0 | 8 | 17 |
| 10 | 12 | 12 | 2 | 8 | 17 |

# SUMMARY AND CONCLUSIONS

We proposed a method of using the knowledge in a network to determine the relevance of individual units. The relevance metric can identify which input or hidden units are most critical to the performance of the network. The least relevant units can then be trimmed to construct a skeleton version of the network.

Skeleton networks have application in two different scenarios, as our simulations demonstrated:

- Understanding the behavior of a network in terms of "rules"

  — *The cue salience problem.* The relevance metric singled out the one input that was sufficient to solve the problem. The other inputs conveyed redundant information.
  — *The rule-plus-exception problem.* The relevance metric was able to distinguish the hidden unit that was responsible for correctly handling most cases (the general rule) from the hidden unit that dealt with an exceptional case.
  — *The train problem.* The relevance metric correctly discovered the minimal set of input features required to describe a category.

- Improving learning performance

  — *The four-bit multiplexor.* Whereas a standard network was often unable to discover a solution, the skeleton network never failed. Further, the skeleton network learned the training set more quickly.
  — *The random mapping problem.* As in the multiplexor problem, the skeleton network succeeded considerably more often with comparable overall learning speed, and less training was required to reach criterion initially.

Basically, the skeletonization technique allows a network to use spare input and hidden units to learn a set of training examples rapidly, and gradually, as units are trimmed, to discover a more concise characterization of the underlying regularities of the task. In the process, local minima seem to be avoided without increasing the overall learning time.

One somewhat surprising result is the ease with which a network is able to recover when a unit is removed. Conventional wisdom has it that if, say, a network is given excess hidden units, it will memorize the training set, thereby making use of all the hidden units available to it. However, in our simulations, the network does not seem to be distributing the solution across all hidden units because even with no further training, removal of a hidden unit often does not drop performance below the criterion. In any case, there generally appears to be an easy path from the solution with many units to the solution with fewer.

Although we have presented skeletonization as a technique for trimming units from a network, there is no reason why a similar procedure could not operate on individual connections instead. Basically, an $\alpha$ coefficient would be required for each connection, allowing for the computation of $\partial E / \partial \alpha$. Yann le Cun (personal communication, 1989) has independently developed a procedure quite similar to our skeletonization technique which operates on individual connections.

## Acknowledgements

Our thanks to Colleen Seifert for conversations that led to this work; to Dave Goldberg, Geoff Hinton, and Yann le Cun for their feedback; and to Eric Jorgensen for saving us from computer hell. This work was supported by grant 87-2-36 from the Sloan Foundation to Geoffrey Hinton, a grant from the James S. McDonnell Foundation to Michael Mozer, and a Sloan Foundation grant and NSF grants IRI-8609599, ECE-8617947, and CDR-8622236 to Paul Smolensky.

## Footnotes

[1] The values reported in Table 1 are an average over 100 replications of the simulation with different initial random weights. Before averaging, however, the signs of the weights were flipped if the hidden-output connection was negative.

[2] Here and below we report median epochs to criterion rather than mean epochs to avoid aberrations caused by the large number of epochs consumed in failure runs.

## References

Chauvin, Y. (1989). A back-propagation algorithm with optimal use of hidden units. In *Advances in Neural Network Information Processing Systems*. San Mateo, CA: Morgan Kaufmann.

Hanson, S. J., & Pratt, L. Y. (1989). Some comparisons of constraints for minimal network construction with back propagation. In *Advances in Neural Network Information Processing Systems*. San Mateo, CA: Morgan Kaufmann.

Medin, D. L., Wattenmaker, W. D., & Michalski, R. S. (1987). Constraints and preferences in inductive learning: An experimental study of human and machine performance. *Cognitive Science, 11*, 299-339.

Mozer, M. C., & Smolensky, P. (1989). *Skeletonization: A technique for trimming the fat from a network via relevance assessment* (Technical Report CU-CS-421-89). Boulder: University of Colorado, Department of Computer Science.

Sejnowski, T. J., & Rosenberg, C. R. (1987). Parallel networks that learn to pronounce English text. *Complex Systems, 1*, 145-168.